# Hierarchical Modeling of Local Image Features through $L_p$-Nested Symmetric Distributions

**Fabian Sinz**
Max Planck Institute for Biological Cybernetics
Spemannstraße 41
72076 Tübingen, Germany
`fabee@tuebingen.mpg.de`

**Eero P. Simoncelli**
Center for Neural Science, and Courant Institute
of Mathematical Sciences, New York University
New York, NY 10003
`eero.simoncelli@nyu.edu`

**Matthias Bethge**
Max Planck Institute for Biological Cybernetics
Spemannstraße 41
72076 Tübingen, Germany
`mbethge@tuebingen.mpg.de`

## Abstract

We introduce a new family of distributions, called $L_p$-*nested symmetric distributions*, whose densities are expressed in terms of a hierarchical cascade of $L_p$-norms. This class generalizes the family of spherically and $L_p$-spherically symmetric distributions which have recently been successfully used for natural image modeling. Similar to those distributions it allows for a nonlinear mechanism to reduce the dependencies between its variables. With suitable choices of the parameters and norms, this family includes the Independent Subspace Analysis (ISA) model as a special case, which has been proposed as a means of deriving filters that mimic complex cells found in mammalian primary visual cortex. $L_p$-nested distributions are relatively easy to estimate and allow us to explore the variety of models between ISA and the $L_p$-spherically symmetric models. By fitting the generalized $L_p$-nested model to $8 \times 8$ image patches, we show that the subspaces obtained from ISA are in fact more dependent than the individual filter coefficients within a subspace. When first applying contrast gain control as preprocessing, however, there are no dependencies left that could be exploited by ISA. This suggests that complex cell modeling can only be useful for redundancy reduction in larger image patches.

## 1 Introduction

Finding a precise statistical characterization of natural images is an endeavor that has concerned research for more than fifty years now and is still an open problem. A thorough understanding of natural image statistics is desirable from an engineering as well as a biological point of view. It forms the basis not only for the design of more advanced image processing algorithms and compression schemes, but also for a better comprehension of the operations performed by the early visual

system and how they relate to the properties of the natural stimuli that are driving it. From both perspectives, redundancy reducing algorithms such as Principal Component Analysis (PCA), Independent Component Analysis (ICA), Independent Subspace Analysis (ISA) and Radial Factorization [11; 21] have received considerable interest since they yield image representations that are favorable for compression and image processing and at the same time resemble properties of the early visual system. In particular, ICA and ISA yield localized, oriented bandpass filters which are reminiscent of receptive fields of simple and complex cells in primary visual cortex [4; 16; 10]. Together with the Redundancy Reduction Hypothesis by Barlow and Attneave [3; 1], those observations have given rise to the idea that these filters represent an important aspect of natural images which is exploited by the early visual system.

Several result, however, show that the density model of ICA is too restricted to provide a good model for natural images patches. Firstly, several authors have demonstrated that filter responses of ICA filters on natural images are not statistically independent [20; 23; 6]. Secondly, after whitening, the optimum of ICA in terms of statistical independence is very shallow or, in other words, all whitening filters yield almost the same redundancy reduction [5; 2]. A possible explanation for that finding is that, after whitening, densities of local image features are approximately spherical [24; 23; 12; 6]. This implies that those densities cannot be made independent by ICA because (i) all whitening filters differ only by an orthogonal transformation, (ii) spherical densities are invariant under orthogonal transformations, and (iii) the only spherical and factorial distribution is the Gaussian. Once local image features become more distant from each other, the contour lines of the density deviates from spherical and become more star-shaped. In order to capture this star-shaped contour lines one can use the more general $L_p$-spherically symmetric distributions which are characterized by densities of the form $\chi(\boldsymbol{y}) = g(\|\boldsymbol{y}\|_p)$ with $\|\boldsymbol{y}\|_p = (\sum |y_i|^p)^{1/p}$ and $p > 0$ [9; 10; 21].

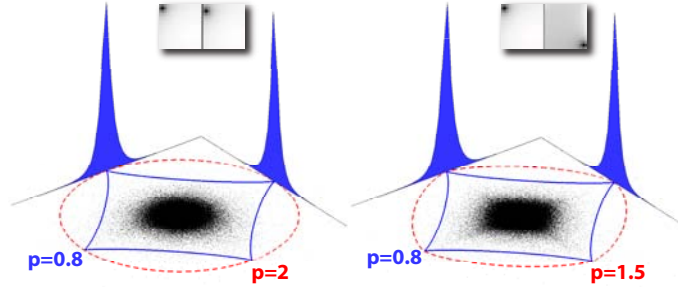

Figure 1: Scatter plots and marginal histograms of neighboring (*left*) and distant (*right*) symmetric whitening filters which are shown at the top. The dashed Contours indicate the unit sphere for the optimal $p$ of the best fitting non-factorial (*dashed line*) and factorial (*solid line*) $L_p$-spherically symmetric distribution, respectively. While close filters exhibit $p = 2$ (spherically symmetric distribution), the value of $p$ decreases for more distant filters.

As illustrated in Figure 1, the relationship between local bandpass filter responses undergoes a gradual transition from $L_2$-spherical for nearby to star-shaped ($L_p$-spherical with $p < 2$) for more distant features [12; 21]. Ultimately, we would expect extremely distant features to become independent, having a factorial density with $p \approx 0.8$. When using a single $L_p$-spherically symmetric model for the joint distribution of nearby and more distant features, a single value of $p$ can only represent a compromise for the whole variety of iso-probability contours. This raises the question whether a combination of local spherical models, as opposed to a single $L_p$-spherical model, yields a better characterization of the statistics of natural image patches. Possible ways to join several local models are Independent Subspace Analysis (ISA) [10], which uses a factorial combination of locally $L_p$-spherical densities, or Markov Random Fields (MRFs) [18; 13]. Since MRFs have the drawback of being implicit density models and computationally very expensive for inference, we will focus on ISA and our model. In principle, ISA could choose its subspaces such that nearby features are grouped into a joint subspace which can then be well described by a spherical symmetric model ($p = 2$) while more distant pixels, living in different subspaces, are assumed to be independent. In fact, previous studies have found ISA to perform better than ICA for image patches as small as $8 \times 8$ and to yield an optimal $p \approx 2$ for the local density models [10]. On the other hand, the ISA model assumes a binary partition into either a $L_p$-spherical or a factorial distribution which does not seem to be fully justified considering the gradual transition described above.

Here, we propose a new family of hierarchical models by replacing the $L_p$-norms in the $L_p$-spherical models by $L_p$-nested functions, which consist of a cascade of nested $L_p$-norms and therefore allow for different values of $p$ for different groups of filters. While this family includes the $L_p$-spherical family and ISA models, it also includes densities that avoid the hard partition into either factorial or $L_p$-spherical. At the same time, parameter estimation for these models can still be similarly efficient and robust as for $L_p$-spherically symmetric models. We find that this family (i) fits the data significantly better than ISA and (ii) generates interesting filters which are grouped in a sensible way within the hierarchy. We also find that, although the difference in performance between $L_p$-spherical and $L_p$-nested models is significant, it is small on $8 \times 8$ patches, suggesting that within this limited spatial range, the iso-probability contours of the joint density can still be reasonably approximated by a single $L_p$-norm. Preliminary results on $16 \times 16$ patches exhibit a more pronounced difference between the $L_p$-nested and the $L_p$-spherically symmetric distribution, suggesting that the change in $p$ becomes more important for modelling densities over a larger spatial range.

## 2   Models

$L_p$-**Nested Symmetric Distributions**    Consider the function

$$f(\boldsymbol{y}) = \left( \left( \sum_{i=1}^{n_1} |y_i|^{p_1} \right)^{\frac{p_\emptyset}{p_1}} + ... + \left( \sum_{i=n_1+...+n_{\ell-1}+1}^{n} |y_i|^{p_\ell} \right)^{\frac{p_\emptyset}{p_\ell}} \right)^{\frac{1}{p_\emptyset}} \tag{1}$$

$$= \left\| \, (\|\boldsymbol{y}_{1:n_1}\|_{p_1}, ..., \|\boldsymbol{y}_{n-n_\ell+1:n}\|_{p_\ell})^\top \, \right\|_{p_\emptyset}.$$

We call this type of functions $L_p$-*nested* and the resulting class of distributions $L_p$-*nested symmetric*. $L_p$-nested symmetric distributions are a special case of the $\nu$-spherical distributions which have a density characterized by the form $\rho(\boldsymbol{y}) = g(\nu(\boldsymbol{y}))$ where $\nu : \mathbb{R}^n \to \mathbb{R}$ is a positively homogeneous function of degree one, i.e. it fulfills $\nu(a\boldsymbol{y}) = a\nu(\boldsymbol{y})$ for any $a \in \mathbb{R}_+$ and $\boldsymbol{y} \in \mathbb{R}^n$ [7]. $L_p$-nested functions are obviously positively homogeneous. Of course, $L_p$-nested functions of $L_p$-nested functions are again $L_p$-nested. Therefore, an $L_p$-nested function $f$ in its general form can be visualized by a tree in which each inner node corresponds to an $L_p$-norm while the leaves stand for the coefficients of the vector $\boldsymbol{y}$.

Because of the positive homogeneity it is possible to normalize a vector $\boldsymbol{y}$ with respect to $\nu$ and obtain a coordinate respresentation $x = r \cdot \boldsymbol{u}$ where $r = \nu(\boldsymbol{y})$ and $\boldsymbol{u} = \boldsymbol{y}/\nu(\boldsymbol{y})$. This implies that the random variable $Y$ has the stochastic representation $Y \doteq RU$ with independent $U$ and $R$ [7] which makes it a generalization of the Gaussian Scale Mixture model [23]. It can be shown that for a given $\nu$, $U$ always has the same distribution while the distribution $\varrho(r)$ of $R$ determines the specific $\rho(\boldsymbol{y})$ [7]. For a general $\nu$, it is difficult to determine the distribution of $U$ since the partition function involves the surface area of the $\nu$-unit sphere which is not analytically tractable in most cases. Here, we show that $L_p$-nested functions allow for an analytical expression of the partition function. Therefore, the corresponding distributions constitute a flexible yet tractable subclass of $\nu$-spherical distributions.

In the remaining paper we adopt the following notational convention: We use multi-indices to index single nodes of the tree. This means that $I = \emptyset$ denotes the root node, $I = (\emptyset, i) = i$ denotes its $i^{th}$ child, $I = (i, j)$ the $j^{th}$ child of $i$ and so on. The function values at individual inner nodes $I$ are denoted by $\mathsf{f}_I$, the vector of function values of the children of an inner node $I$ by $\mathbf{f}_{I,1:\ell_I} = (\mathsf{f}_{I,1}, ..., \mathsf{f}_{I,\ell_I})^\top$. By definition, parents and children are related via $\mathsf{f}_I = \|\mathbf{f}_{I,1:\ell_I}\|_{p_I}$. The number of children of a particular node $I$ is denoted by $\ell_I$.

$L_p$-nested symmetric distributions are a very general class of densities. For instance, since every $L_p$-norm $\| \cdot \|_p$ is an $L_p$-nested function, $L_p$-nested distributions includes the family of $L_p$-spherically symmetric distributions including (for $p = 2$) the family of spherically symmetric distributions. When e.g. setting $f = \| \cdot \|_2$ or $f = (\| \cdot \|_2^p)^{1/p}$, and choosing the radial distribution $\varrho$ appropriately, one can recover the Gaussian $\rho(\boldsymbol{y}) = Z^{-1} \exp\left(-\|\boldsymbol{y}\|_2^2\right)$ or the generalized spherical Gaussian $\rho(\boldsymbol{y}) = Z^{-1} \exp\left(-\|\boldsymbol{y}\|_2^p\right)$, respectively. On the other hand, when choosing the $L_p$-nested function $f$ as in equation (1) and $\varrho$ to be the radial distribution of a $p$-generalized Normal distribution $\varrho(r) =$

$Z^{-1} r^{n-1} \exp\left(-r^{p_\emptyset}/s\right)$ [8; 22], the inner nodes $\mathbf{f}_{1:\ell_\emptyset}$ become independent and we can recover an ISA model. Note, however, that not all ISA models are also $L_p$-nested since $L_p$-nested symmetry requires the radial distribution to be that of a $p$-generalized Normal.

In general, for a given radial distribution $\varrho$ on the $L_p$-nested radius $f(\boldsymbol{y})$, an $L_p$-nested symmetric distribution has the form

$$\rho(\boldsymbol{y}) = \frac{1}{\mathcal{S}_f(f(\boldsymbol{y}))} \cdot \varrho(f(\boldsymbol{y})) = \frac{1}{\mathcal{S}_f(1) \cdot f^{n-1}(\boldsymbol{y})} \cdot \varrho(f(\boldsymbol{y})) \tag{2}$$

where $\mathcal{S}_f(f(\boldsymbol{y})) = \mathcal{S}_f(1) \cdot f^{n-1}(\boldsymbol{y})$ is the surface area of the $L_p$-nested sphere with the radius $f(\boldsymbol{y})$. This means that the partition function of a general $L_p$-nested symmetric distribution is the partition function of the radial distribution normalized by the surface area of the $L_p$-nested sphere with radius $f(\boldsymbol{y})$. For a given $f$ and a radius $\mathsf{f}_\emptyset = f(\boldsymbol{y})$ this surface area is given by the equation

$$\mathcal{S}_f(\mathsf{f}_\emptyset) = \mathsf{f}_\emptyset^{n-1} 2^n \prod_{I \in \mathcal{I}} \frac{1}{p_I^{\ell_I-1}} \prod_{k=1}^{\ell_I-1} B\left[\frac{\sum_{i=1}^k n_{I,k}}{p_I}, \frac{n_{I,k+1}}{p_I}\right] = \mathsf{f}_\emptyset^{n-1} 2^n \prod_{I \in \mathcal{I}} \frac{\prod_{k=1}^{\ell_I} \Gamma\left[\frac{n_{I,k}}{p_I}\right]}{p_I^{\ell_I-1} \Gamma\left[\frac{n_I}{p_I}\right]}$$

where $\mathcal{I}$ denotes the set of all multi-indices of inner nodes, $n_I$ the number of leaves of the subtree under $I$ and $B[a,b]$ the beta function. Therefore, if the partition function of the radial distribution can be computed easily, so can the partition function of the multivariate $L_p$-nested distribution.

Since the only part of equation (2) that includes free parameters is the radial distribution $\varrho$, maximum likelihood estimation of those parameters $\boldsymbol{\vartheta}$ can be carried out on the univariate distribution $\varrho$ only, because

$$\mathrm{argmax}_{\boldsymbol{\vartheta}} \log \rho(\boldsymbol{y}|\boldsymbol{\vartheta}) \overset{(2)}{=} \mathrm{argmax}_{\boldsymbol{\vartheta}} \left(-\log \mathcal{S}_f(f(\boldsymbol{y})) + \log \varrho(f(\boldsymbol{y})|\boldsymbol{\vartheta})\right) = \mathrm{argmax}_{\boldsymbol{\vartheta}} \log \varrho(f(\boldsymbol{y})|\boldsymbol{\vartheta}).$$

This means that parameter estimation can be done efficiently and robustly on the values of the $L_p$-nested function.

Since, for a given $f$, an $L_p$-nested distribution is fully specified by a radial distribution, changing the radial distribution also changes the $L_p$-nested distribution. This suggests an image decomposition constructed from a cascade of nonlinear, gain-control-like mappings reducing the dependence between the filter coefficients. Similar to Radial Gaussianization or $L_p$-Radial Factorization algorithms [12; 21], the radial distribution $\varrho_\emptyset$ of the root node is mapped into the radial distribution of a $p$-generalized Normal via histogram equalization, thereby making its children exponential power distributed and statistically independent [22]. This procedure is then repeated recursively for each of the children until the leaves of the tree are reached.

Below, we estimate the multi-information (MI) between the filters or subtrees at different levels of the hierarchy. In order to do that robustly, we need to know the joint distribution of their values. In particular, we are interested in the joint distribution of the children $\mathbf{f}_{I,1:\ell_I}$ of a node $I$ (e.g. layer 2 in Figure 2). Just from the form of an $L_p$-nested function one might guess that those children are $L_p$-spherically symmetric distributed. However, this is not the case. For example, the children $\mathbf{f}_{1:\ell_\emptyset}$ of the root node (assuming that none of them is a leaf) follow the distribution

$$\rho(\mathbf{f}_{1:\ell_\emptyset}) = \frac{\varrho_\emptyset(\|\mathbf{f}_{1:\ell_\emptyset}\|_{p_\emptyset})}{S_{\|\cdot\|_{p_\emptyset}}(\|\mathbf{f}_{1:\ell_\emptyset}\|_{p_\emptyset})} \prod_{i=1}^{\ell_\emptyset} \mathsf{f}_i^{n_i-1}. \tag{3}$$

This implies that $\mathbf{f}_{1:\ell_\emptyset}$ can be represented as a product of two independent random variables $\boldsymbol{u} = \mathbf{f}_{1:\ell_\emptyset}/\|\mathbf{f}_{1:\ell_\emptyset}\|_{p_\emptyset} \in \mathbb{R}_+^{\ell_\emptyset}$ and $r = \|\mathbf{f}_{1:\ell_\emptyset}\|_{p_\emptyset} \in \mathbb{R}_+$ with $r \sim \varrho_\emptyset$ and $\left(u_1^{p_\emptyset}, ..., u_{\ell_\emptyset}^{p_\emptyset}\right) \sim$ Dir $\left[n_1/p_\emptyset, ..., n_{\ell_\emptyset}/p_\emptyset\right]$ following a Dirichlet distribution (see Additional Material). We call this distribution a *Dirichlet Scale Mixture (DSM)*. A similar form can be shown for the joint distribution of leaves and inner nodes (summarizing the whole subtree below them). Unfortunately, only the children $\mathbf{f}_{1:\ell_\emptyset}$ of the root node are really DSM distributed. We were not able to analytically calculate the marginal distribution of an arbitrary node's children $\mathbf{f}_{I,1:\ell_I}$, but we suspect it to have a similar form. For that reason we fit DSMs to those children $\mathbf{f}_{I,1:\ell_\emptyset}$ in the experiments below and use the estimated model to assess the dependencies between them. We also use it for measuring the dependencies between the subspaces of ISA.

Fitting DSMs via maximum likelihood can be carried out similarly to estimating $L_p$-nested distributions: Since the radial variables $\boldsymbol{u}$ and $r$ are independent, the Dirichlet and the radial distribution can be estimated on the normalized data points $\{\boldsymbol{u}_i\}_{i=1}^m$ and their respective norms $\{r_i\}_{i=1}^m$ independently.

**$L_p$-Spherically Symmetric Distributions and Independent Subspace Analysis**   The family of $L_p$-spherically symmetric distributions are a special case of $L_p$-nested distributions for which $f(\boldsymbol{y}) = \|\boldsymbol{y}\|_p$ [9]. We use the ISA model by [10] where the filter responses $\boldsymbol{y}$ are modelled by a factorial combination of $L_p$-spherically symmetric distributions sitting on each subspace

$$\rho(\boldsymbol{y}) = \prod_{k=1}^K \rho_k(\|\boldsymbol{y}_{I_k}\|_{p_k}).$$

## 3   Experiments

Given an image patch $\boldsymbol{x}$, all models used in this paper define densities over filter responses $\boldsymbol{y} = W\boldsymbol{x}$ of linear filters. This means, that all models have the form $\rho(\boldsymbol{y}) = |\det W| \cdot \rho(W\boldsymbol{x})$. The $(n-1) \times n$ matrix $W$ has the form $W = QSP$ where $P \in \mathbb{R}^{(n-1) \times n}$ has mutually orthogonal rows and projects onto the orthogonal complement of the DC-filter (filter with equal coefficients), $S \in \mathbb{R}^{(n-1) \times (n-1)}$ is a whitening matrix and $Q \in SO_{n-1}$ is an orthogonal matrix determining the final filter shapes of $W$. When we speak of optimizing the filters according to a model, we mean optimizing $Q$ over $SO_{n-1}$. The reason for projecting out the DC component is, that it can behave quite differently depending on the dataset. Therefore, it is usually removed and modelled separately. Since the DC component is the same for all models and would only add a constant offset to the measures we use in our experiments, we ignore it in the experiments below.

**Data**   We use ten pairs of independently sampled training and test sets of $8 \times 8$ ($16 \times 16$) patches from the van Hateren dataset, each containing $100,000$ ($500,000$) examples. Hyvärinen and Köster [10] report that ISA already finds several subspaces for $8 \times 8$ image patches. We perform all experiments with two different types of preprocessing: either we only whiten the data (WO-data), or we whiten it and apply an additional contrast gain control step (CGC-data), for which we use the radial factorization method described in [12; 21] with $p = 2$ in the symmetric whitening basis.

We use the same whitening procedure as in [21; 6]: Each dataset is centered on the mean over examples and dimensions and rescaled such that whitening becomes volume conserving. Similarly, we use the same orthogonal matrix to project out the DC-component of each patch (matrix $P$ above). On the remaining $n-1$ dimensions, we perform symmetric whitening (SYM) with $S = C^{-\frac{1}{2}}$ where $C$ denotes the covariance matrix of the DC-corrected data $C = \text{cov}\,[PX]$.

**Evaluation Measures**   We use the *Average Log Loss* per component (ALL) for assessing the quality of the different models, which we estimate by taking the empirical average over a large ensemble of test points $ALL = -\frac{1}{n-1}\left\langle \log \rho(\boldsymbol{y}) \right\rangle_Y \approx -\frac{1}{m(n-1)} \sum_{i=1}^m \log \rho(\boldsymbol{y}_i)$. The ALL equals the entropy if the model distribution equals the true distribution and is larger otherwise. For the CGC-data, we adjust the ALL by the log-determinant of the CGC transformation [11]. In contrast to [10] this allows us to quantitively compare models across the two different types of preprocessing (WO and CGC), which was not possible in [10].

In order to measure the dependence between different random variables, we use the *multi-information* per component (MI) $\frac{1}{n-1}\left(\sum_{i=1}^d H[Y_i] - H[Y]\right)$ which is the difference between the sum of the marginal entropies and the joint entropy. The MI is a positive quantity which is zero if and only if the joint distribution is factorial. We estimate the marginal entropies by a jackknifed MLE entropy estimator [17] (corrected for the log of the bin width in order to estimate the differential entropy) where we adjust the bin width of the histograms suggested by Scott [19]. Instead of the joint entropy, we use the ALL of an appropriate model distribution. Since the ALL is theoretically always larger than the true joint entropy (ignoring estimation errors) using the ALL instead of the joint entropy should underestimate the true MI, which is still sufficient for our purpose.

**Parameter Estimation**   For all models (ISA, DSM, $L_p$-spherical and $L_p$-nested), we estimate the parameters $\boldsymbol{\vartheta}$ for the radial distribution as described above in Section 2. For a given filter matrix

$W$ the values of the exponents $p$ are estimated by minimizing the ALL at the ML estimates $\hat{\boldsymbol{\vartheta}}$ over $\boldsymbol{p} = (p_1, ..., p_q)^\top$. For the $L_p$-nested distributions, we use the Nelder-Mead [15] method for the optimization over $\boldsymbol{p} = (p_1, ..., p_q)^\top$ and for the $L_p$-spherically symmetric distributions we use Golden Search over the single $p$. For the ISA model, we carry out a Golden Search over $p$ for each subspace independently. For the $L_p$-spherical and the single models on the ISA subspaces, we use a search range of $p \in [0.1, 2.1]$ on $p$. For estimating the Dirichlet Scale Mixtures, we use the `fastfit` package by Tom Minka to estimate the parameters of the Dirichlet distribution. The radial distribution is estimated independently as described above.

When fitting the filters $W$ to the different models (ISA, $L_p$-spherical and $L_p$-nested), we use a gradient ascent on the log-likelihood over the orthogonal group by alternating between optimizing the parameters $\mathbf{p}$ and $\vartheta$ and optimizing for $W$. For the gradient ascent, we compute the standard Euclidean gradient with respect to $W \in \mathbb{R}^{(n-1)\times(n-1)}$ and project it back onto the tangent space of $SO_{n-1}$. Using the gradient $\nabla W$ obtained in that manner, we perform a line search with respect to $t$ using the backprojections of $W + t \cdot \nabla W$ onto $SO_{n-1}$. This method is a simplified version of the one proposed by [14].

**Experiments with Independent Subspace Analysis and $L_p$-Spherically Symmetric Distributions** We optimized filters for ISA models with $K = 2, 4, 8, 16$ subspaces embracing $32, 16, 8, 4$ components (one subspace always had one dimension less due to the removal of the DC component), and for an $L_p$-spherically symmetric model. When optimizing for $W$ we use a radial $\Gamma$-distribution for the $L_p$-spherically symmetric models and a radial $\chi^p$ distribution ($\|\boldsymbol{y}_{I_k}\|_{p_k}^{p_k}$ is $\Gamma$-distributed) for the models on the single single subspaces of ISA, which is closer to the one used by [10]. After optimization, we make a final optimization for $\boldsymbol{p}$ and $\boldsymbol{\vartheta}$ using a mixture of log normal distributions ($\log \mathcal{N}$) with $K = 6$ mixture components on the radial distribution(s).

**$L_p$-Nested Symmetric Distributions** As for the $L_p$-spherically symmetric models, we use a radial $\Gamma$-distribution for the optimization of $W$ and a mixture of $\log \mathcal{N}$ distributions for the final fit. We use two different kind of tree structures for our experiments with $L_p$-nested symmetric distributions. In the *deep tree* (DT) structure we first group $2 \times 2$ blocks of four neighboring SYM filters. Afterwards, we group those blocks again in a quadtree manner until we reached the root node (see Figure 2**A**). The second tree structure (PND$_k$) was motivated by ISA. Here, we simply group the filter within each subspace and joined them at the root node afterwards (see Figure 2**B**). In order to speed up parameter estimation, each layer of the tree shared the same value of $p$.

**Multi-Information Measurements** For the ISA models, we estimated the MI between the filter responses within each subspace and between the $L_p$-radii $\|\boldsymbol{y}_{I_k}\|_{p_k}, 1 \le k \le K$. In the former case we used the ALL of an $L_p$-spherically symmetric distribution with especially optimized $p$ and $\boldsymbol{\vartheta}$, in the latter a DSM with optimized radial and Dirichlet distribution as a surrogate for the joint entropy. For the $L_p$-nested distribution, we estimate the MI between the children $\mathbf{f}_{I,1:\ell_I}$ of all inner nodes $I$. In case the children are leaves, we use the ALL of an $L_p$-spherically symmetric distribution as surrogate for the joint entropy, in case the children are inner nodes themselves, we use the ALL of an DSM. The red arrows in Figure 2**A** exemplarily depict the entities between which the MI was estimated.

## 4   Results and Discussion

Figure (2) shows the optimized filters for the DT and the PND$_{16}$ tree structure (we included the filters optimized on the first of ten datasets for all tree structures in the Additional Material). For both tree structures, the filters on the lowest level are grouped according to spatial frequency and orientation, whereas the variation in orientation is larger for the PND$_{16}$ tree structure and some filters are unoriented. The next layer of inner nodes, which is only present in the DT tree structure, roughly joins spatial location, although each of those inner nodes has one child whose leaves are global filters.

When looking at the various values of $p$ at the inner nodes, we can observe that nodes which are higher up in the tree usually exhibit a smaller value of $p$. Surprisingly, as can be seen in Figure 3 **B** and **C**, a smaller value of $p$ does not correspond to a larger independence between the subtrees, which are even more correlated because almost every subtree contains global filters. The small value of $p$ is caused by the fact that the DSM (the distribution of the subtree values) has to account for this correlation which it can only do by decreasing the value of $p$ (see Figure 3 and the DSM in

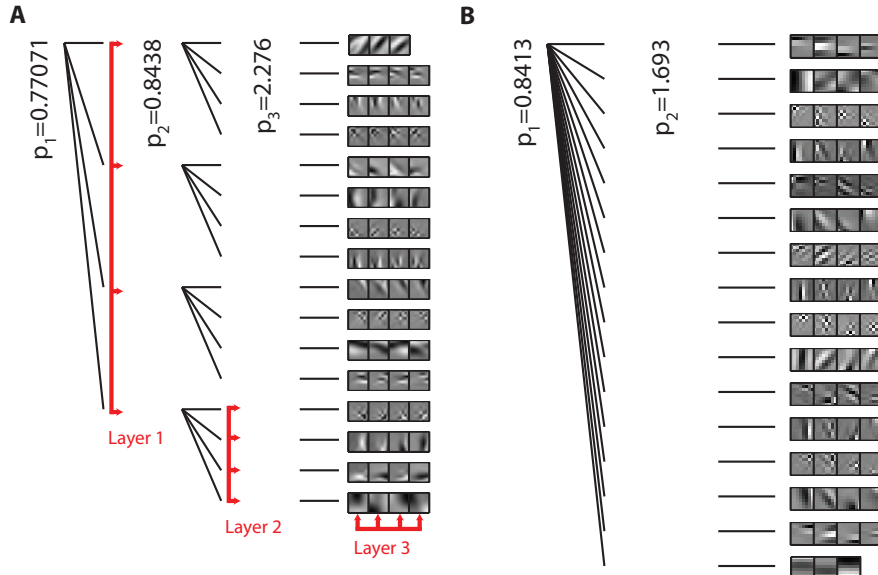

Figure 2: **Examples for the tree structures of $L_p$-nested distributions used in the experiments:** (**A**) shows the DT structure with the corresponding optimized values. The red arrows display examples of groups of filters or inner nodes, respectively, for which we estimated the MI. (**B**) shows the $\text{PND}_{16}$ tree structure with the corresponding values of $p$ at the inner nodes and the optimized filters.

the Additional Material). Note that this finding is exactly opposite to the assumptions in the ISA model which can usually not generate such a behavior (Figure 3**A**) as it models the two subtrees to be independent. This is likely to be one reason for the higher ALL of the ISA models (see Table 1).

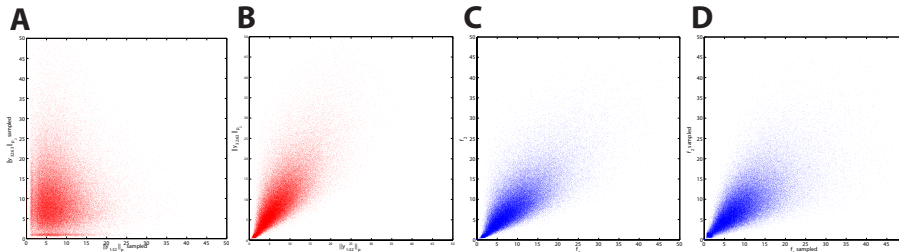

Figure 3: Independence of subspaces for WO-data not justfied: (**A**) Subspace radii sampled from $\text{ISA}_2$, (**B**) subspace radii of natural image patches in the $\text{ISA}_2$ basis, (**C**) subtree values of the $\text{PND}_2$ in the $\text{PND}_2$ basis, and (**D**) samples from the $\text{PND}_2$ model. While the $\text{ISA}_2$ model spreads out the radii almost over the whole positive quadrant due to the independence assumption the samples from the $L_p$-nested subtrees are more concentrated around the diagonal like the true data. The $L_p$-nested model can achieve this behavior since (i) it does not assume a radial distribution that leads to independent radii on the subtrees and (ii) the subtree values $f_1$ and $f_2$ are $\text{DSM}[n_1/p_\emptyset, n_2/p_\emptyset,]$ distributed. By changing the value of $p_\emptyset$, the DSM model can put more mass towards the diagonal, which produces the "beam-like" behavior shown in the plot.

Table 1 shows the ALL and the MI measurements for all models. Except for the ISA models on WO-data, all performances are similar, whereas the $L_p$-nested models usually achieve the lowest ALL independent of the particular tree structure used. For the WO-data, the $L_p$-spherical and the $\text{ISA}_2$ model come close to the performance of the $L_p$-nested models. For the other ISA models on WO-data the ALL gets worse with increasing number of subspaces (bold font number in Table 1). This reflects the effect described above: Contrary to the assumptions of the ISA model, the responses of the different subspaces become in fact more correlated than the single filter responses. This can also be seen in the MI measurements discussed below.

When looking at the ALL for CGC data, on the other hand, ISA suddenly becomes competitive. This importance of CGC for ISA has already been noted in [10]. The small differences between all the models in the CGC case shows that the contour change of the joint density for $8 \times 8$ patches is too small to allow for a large advantage of the $L_p$-nested model, because contrast gain control (CGC)

directly corresponds to modeling the distribution with an $L_p$-spherically symmetric distribution [21]. Preliminary results on $16 \times 16$ data ($1.39 \pm 0.003$ for the $L_p$-nested and $1.45 \pm 0.003$ for the $L_p$-spherical model on WO-data), however, show a more pronounced improvement with for the $L_p$-nested model, indicating that a single $p$ does not suffice anymore to capture all dependencies when going to larger patch sizes.

When looking at the MI measurements between the filters/subtrees at different levels of the hierarchy in the $L_p$-nested, $L_p$-spherically symmetric and ISA models, we can observe that for the WO-data, the MI actually increases when going from lower to higher layers. This means that the MI between the direct filter responses (layer 3 for DT and layer 2 for all others) is in fact lower than the MI between the subspace radii or the inner nodes of the $L_p$-nested tree (layer 1-2 for DT, layer 1 for all others). The highest MI is achieved between the children of the root node for the DT tree structure (DT layer 1). As explained above this observation contradicts the assumptions of the ISA model and probably causes it worse performance on the WO-data.

For the CGC-data, on the other hand, the MI has been substantially decreased by CGC over all levels of the hierarchy. Furthermore, the single filter responses inside a particular subspace or subtree are now more dependent than the subtrees or subspaces themselves. This suggests that the competitive performance of ISA is not due to the model but only due to the fact that CGC made the data already independent. In order to double check this result, we fitted an ICA model to the CGC-data [21] and found an ALL of $1.41 \pm 0.004$ which is very close to the performance of ISA and the $L_p$-nested distributions (which would not be the case for WO-data [21]).

Taken together, the ALL and the MI measurements suggest that ISA is not the best way to join multiple local models into a single joint model. The basic assumption of the ISA model for natural images is that filter coefficients can either be dependent within a subspace or must be independent between different subspaces. However, the increasing ALL for an increasing number of subspaces and the fact that the MI between subspaces is actually higher than within the subspaces, demonstrates that this hard partition is not justified when the data is only whitened.

| Family | $L_p$-nested | | | | |
|---|---|---|---|---|---|
| **Model** | Deep Tree | $PND_2$ | $PND_4$ | $PND_8$ | $PND_{16}$ |
| **ALL** | $1.39 \pm 0.004$ | $1.39 \pm 0.004$ | $1.39 \pm 0.004$ | $1.40 \pm 0.004$ | $1.39 \pm 0.004$ |
| **ALL CGC** | $1.39 \pm 0.005$ | $1.40 \pm 0.004$ | $1.40 \pm 0.005$ | $1.40 \pm 0.004$ | $1.39 \pm 0.004$ |
| **MI Layer 1** | $0.84 \pm 0.019$ | $0.48 \pm 0.008$ | $0.7 \pm 0.002$ | $0.75 \pm 0.003$ | $0.61 \pm 0.0036$ |
| **MI Layer 1 CGC** | $0.0 \pm 0.004$ | $0.10 \pm 0.002$ | $0.02 \pm 0.003$ | $0.0 \pm 0.009$ | $0.0 \pm 0.01$ |
| **MI Layer 2** | $0.42 \pm 0.021$ | $0.35 \pm 0.017$ | $0.33 \pm 0.017$ | $0.28 \pm 0.019$ | $0.25 \pm 0.025$ |
| **MI Layer 2 CGC** | $0.002 \pm 0.005$ | $0.01 \pm 0.0008$ | $0.01 \pm 0.004$ | $0.01 \pm 0.006$ | $0.02 \pm 0.008$ |
| **MI Layer 3** | $0.28 \pm 0.036$ | - | - | - | - |
| **MI Layer 3 GCG** | $0.04 \pm 0.005$ | - | - | - | - |
| Family | $L_p$-spherical | ISA | | | |
| **Model** | - | $ISA_2$ | $ISA_4$ | $ISA_8$ | $ISA_{16}$ |
| **ALL** | $1.41 \pm 0.004$ | $\mathbf{1.40 \pm 0.005}$ | $\mathbf{1.43 \pm 0.006}$ | $\mathbf{1.46 \pm 0.006}$ | $\mathbf{1.55 \pm 0.006}$ |
| **ALL CGC** | $1.41 \pm 0.004$ | $1.41 \pm 0.008$ | $1.39 \pm 0.007$ | $1.40 \pm 0.005$ | $1.41 \pm 0.007$ |
| **MI Layer 1** | $0.34 \pm 0.004$ | $0.47 \pm 0.01$ | $0.69 \pm 0.012$ | $0.7 \pm 0.018$ | $0.63 \pm 0.0039$ |
| **MI Layer 1 CGC** | $0.00 \pm 0.005$ | $0.00 \pm 0.09$ | $0.00 \pm 0.06$ | $0.00 \pm 0.04$ | $0.00 \pm 0.02$ |
| **MI Layer 2** | - | $0.36 \pm 0.017$ | $0.33 \pm 0.019$ | $0.31 \pm 0.032$ | $0.24 \pm 0.024$ |
| **MI Layer 2 CGC** | - | $0.004 \pm 0.003$ | $0.03 \pm 0.012$ | $0.02 \pm 0.018$ | $0.0006 \pm 0.013$ |

Table 1: **ALL and MI for all models:** The upper part shows the results for the $L_p$-nested models, the lower part show the results for the $L_p$-spherical and the ISA models. The ALL for the $L_p$-nested models is almost equal for all tree structures and a bit lower compared to the $L_p$-spherical and the ISA models. For the whitened only data, the ALL increases significantly with the number of subspaces (**bold** font). For the CGC data, most models perform similarly well. When looking at the MI, we can see that higher layers for whitened only data are in fact more dependent than lower ones. For CGC data, the MI has dropped substantially over all layers due to CGC. In that case, the lower layers are more independent.

In summary, our results show that $L_p$-nested symmetric distributions yield a good performance on natural image patches, although the advantage over $L_p$-spherically symmetric distributions is fairly small, suggesting that the distribution within these small patches ($8 \times 8$) is captured reasonably well by a single $L_p$-norm. Furthermore, our results demonstrate that—at least for $8 \times 8$ patches—the assumptions of ISA are too rigid for WO-data and are trivially fulfilled for the CGC-data, since CGC already removed most of the dependencies. We are currently working to extend this study to larger patches, which we expect will reveal a more significant advantage for $L_p$-nested models.

# References

[1] F. Attneave. Informational aspects of visual perception. *Psychological Review*, 61:183–193, 1954.

[2] R. Baddeley. Searching for filters with "interesting" output distributions: an uninteresting direction to explore? *Network: Computation in Neural Systems*, 7(2):409–421, 1996.

[3] H. B. Barlow. *Sensory mechanisms, the reduction of redundancy, and intelligence*. 1959.

[4] Anthony J. Bell and Terrence J. Sejnowski. An Information-Maximization approach to blind separation and blind deconvolution. *Neural Computation*, 7(6):1129–1159, November 1995.

[5] Matthias Bethge. Factorial coding of natural images: how effective are linear models in removing higher-order dependencies? *Journal of the Optical Society of America A*, 23(6):1253–1268, June 2006.

[6] Jan Eichhorn, Fabian Sinz, and Matthias Bethge. Natural image coding in v1: How much use is orientation selectivity? *PLoS Comput Biol*, 5(4):e1000336, April 2009.

[7] Carmen Fernandez, Jacek Osiewalski, and Mark F. J. Steel. Modeling and inference with $\nu$-spherical distributions. *Journal of the American Statistical Association*, 90(432):1331–1340, Dec 1995.

[8] Irwin R. Goodman and Samuel Kotz. Multivariate $\theta$-generalized normal distributions. *Journal of Multivariate Analysis*, 3(2):204–219, Jun 1973.

[9] A. K. Gupta and D. Song. $l_p$-norm spherical distribution. *Journal of Statistical Planning and Inference*, 60:241–260, 1997.

[10] A. Hyvarinen and U. Koster. Complex cell pooling and the statistics of natural images. *Network: Computation in Neural Systems*, 18(2):81–100, 2007.

[11] S Lyu and E P Simoncelli. Nonlinear extraction of 'independent components' of natural images using radial Gaussianization. *Neural Computation*, 21(6):1485–1519, June 2009.

[12] S Lyu and E P Simoncelli. Reducing statistical dependencies in natural signals using radial Gaussianization. In D. Koller, D. Schuurmans, Y. Bengio, and L. Bottou, editors, *Adv. Neural Information Processing Systems 21*, volume 21, pages 1009–1016, Cambridge, MA, May 2009. MIT Press.

[13] Siwei Lyu and E.P. Simoncelli. Modeling multiscale subbands of photographic images with fields of gaussian scale mixtures. *Pattern Analysis and Machine Intelligence, IEEE Transactions on*, 31(4):693–706, 2009.

[14] J. H. Manton. Optimization algorithms exploiting unitary constraints. *IEEE Transactions on Signal Processing*, 50:635 – 650, 2002.

[15] J. A. Nelder and R. Mead. A simplex method for function minimization. *The Computer Journal*, 7(4):308–313, Jan 1965.

[16] Bruno A. Olshausen and David J. Field. Emergence of simple-cell receptive field properties by learning a sparse code for natural images. *Nature*, 381(6583):607–609, June 1996.

[17] Liam Paninski. Estimation of entropy and mutual information. *Neural Computation*, 15(6):1191–1253, Jun 2003.

[18] S. Roth and M.J. Black. Fields of experts: a framework for learning image priors. In *Computer Vision and Pattern Recognition, 2005. CVPR 2005. IEEE Computer Society Conference on*, volume 2, pages 860–867 vol. 2, 2005.

[19] David W. Scott. On optimal and data-based histograms. *Biometrika*, 66(3):605–610, Dec 1979.

[20] E.P. Simoncelli. Statistical models for images: compression, restoration and synthesis. In *Signals, Systems & Computers, 1997. Conference Record of the Thirty-First Asilomar Conference on*, volume 1, pages 673–678 vol.1, 1997.

[21] F. Sinz and M. Bethge. The conjoint effect of divisive normalization and orientation selectivity on redundancy reduction. In *Neural Information Processing Systems 2008*, 2009.

[22] F. H. Sinz, S. Gerwinn, and M. Bethge. Characterization of the p-generalized normal distribution. *Journal of Multivariate Analysis*, 100(5):817–820, 05 2009.

[23] M. J. Wainwright and E. P. Simoncelli. Scale mixtures of gaussians and the statistics of natural images. In *Advances in neural information processing systems*, volume 12, pages 855–861, 2000.

[24] Christoph Zetzsche, Gerhard Krieger, and Bernhard Wegmann. The atoms of vision: Cartesian or polar? *Journal of the Optical Society of America A*, 16(7):1554–1565, Jul 1999.

